# Constraining a Bayesian Model of Human Visual Speed Perception

**Alan A. Stocker**  and **Eero P. Simoncelli**
Howard Hughes Medical Institute,
Center for Neural Science, and Courant Institute of Mathematical Sciences
New York University, U.S.A.

## Abstract

It has been demonstrated that basic aspects of human visual motion perception are qualitatively consistent with a Bayesian estimation framework, where the prior probability distribution on velocity favors slow speeds. Here, we present a refined probabilistic model that can account for the typical trial-to-trial variabilities observed in psychophysical speed perception experiments. We also show that data from such experiments can be used to constrain both the likelihood and prior functions of the model. Specifically, we measured matching speeds and thresholds in a two-alternative forced choice speed discrimination task. Parametric fits to the data reveal that the likelihood function is well approximated by a LogNormal distribution with a characteristic contrast-dependent variance, and that the prior distribution on velocity exhibits significantly heavier tails than a Gaussian, and approximately follows a power-law function.

Humans do not perceive visual motion veridically. Various psychophysical experiments have shown that the perceived speed of visual stimuli is affected by stimulus contrast, with low contrast stimuli being perceived to move slower than high contrast ones [1, 2]. Computational models have been suggested that can qualitatively explain these perceptual effects. Commonly, they assume the perception of visual motion to be *optimal* either within a deterministic framework with a regularization constraint that biases the solution toward zero motion [3, 4], or within a probabilistic framework of Bayesian estimation with a prior that favors slow velocities [5, 6].

The solutions resulting from these two frameworks are similar (and in some cases identical), but the probabilistic framework provides a more principled formulation of the problem in terms of meaningful probabilistic components. Specifically, Bayesian approaches rely on a likelihood function that expresses the relationship between the noisy measurements and the quantity to be estimated, and a prior distribution that expresses the probability of encountering any particular value of that quantity. A probabilistic model can also provide a richer description, by defining a full probability density over the set of possible "percepts", rather than just a single value. Numerous analyses of psychophysical experiments have made use of such distributions within the framework of signal detection theory in order to model perceptual behavior [7].

Previous work has shown that an ideal Bayesian observer model based on Gaussian forms

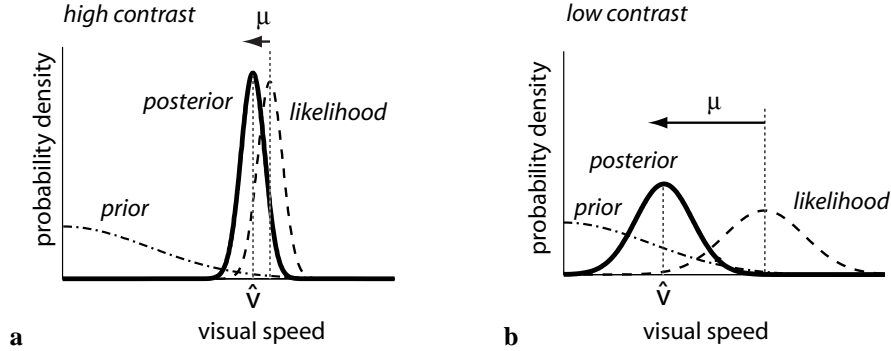

Figure 1: *Bayesian model of visual speed perception.* a) For a high contrast stimulus, the likelihood has a narrow width (a high signal-to-noise ratio) and the prior induces only a small shift $\mu$ of the mean $\hat{v}$ of the posterior. b) For a low contrast stimuli, the measurement is noisy, leading to a wider likelihood. The shift $\mu$ is much larger and the perceived speed lower than under condition (a).

for both likelihood and prior is sufficient to capture the basic qualitative features of global translational motion perception [5, 6]. But the behavior of the resulting model deviates systematically from human perceptual data, most importantly with regard to trial-to-trial variability and the precise form of interaction between contrast and perceived speed. A recent article achieved better fits for the model under the assumption that human contrast perception saturates [8]. In order to advance the theory of Bayesian perception and provide significant constraints on models of neural implementation, it seems essential to constrain quantitatively both the likelihood function and the prior probability distribution. In previous work, the proposed likelihood functions were derived from the brightness constancy constraint [5, 6] or other generative principles [9]. Also, previous approaches defined the prior distribution based on general assumptions and computational convenience, typically choosing a Gaussian with zero mean, although a Laplacian prior has also been suggested [4]. In this paper, we develop a more general form of Bayesian model for speed perception that can account for trial-to-trial variability. We use psychophysical speed discrimination data in order to constrain both the likelihood and the prior function.

# 1    Probabilistic Model of Visual Speed Perception

## 1.1    Ideal Bayesian Observer

Assume that an observer wants to obtain an estimate for a variable $v$ based on a measurement $m$ that she/he performs. A Bayesian observer "knows" that the measurement device is not ideal and therefore, the measurement $m$ is affected by noise. Hence, this observer combines the information gained by the measurement $m$ with *a priori* knowledge about $v$. Doing so (and assuming that the prior knowledge is valid), the observer will – on average – perform better in estimating $v$ than just trusting the measurements $m$. According to Bayes' rule

$$p(v|m) = \frac{1}{\alpha}p(m|v)p(v) \tag{1}$$

the probability of perceiving $v$ given $m$ (*posterior*) is the product of the *likelihood* of $v$ for a particular measurements $m$ and the *a priori* knowledge about the estimated variable $v$ (*prior*). $\alpha$ is a normalization constant independent of $v$ that ensures that the posterior is a proper probability distribution.

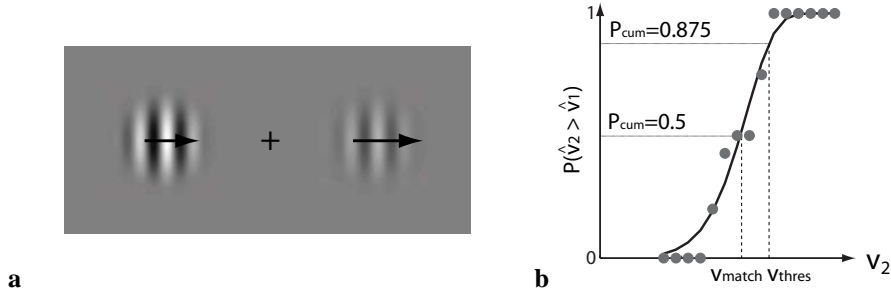

Figure 2: *2AFC speed discrimination experiment.* a) Two patches of drifting gratings were displayed simultaneously (motion without movement). The subject was asked to fixate the center cross and decide after the presentation which of the two gratings was moving faster. b) A typical psychometric curve obtained under such paradigm. The dots represent the empirical probability that the subject perceived stimulus2 moving faster than stimulus1. The speed of stimulus1 was fixed while $v_2$ is varied. The point of subjective equality, $v_{match}$, is the value of $v_2$ for which $P_{cum} = 0.5$. The threshold velocity $v_{thresh}$ is the velocity for which $P_{cum} = 0.875$.

It is important to note that the measurement $m$ is an internal variable of the observer and is not necessarily represented in the same space as $v$. The likelihood embodies both the mapping from $v$ to $m$ and the noise in this mapping. So far, we assume that there is a monotonic function $f(v) : v \rightarrow v_m$ that maps $v$ into the same space as $m$ (m-space). Doing so allows us to analytically treat $m$ and $v_m$ in the same space. We will later propose a suitable form of the mapping function $f(v)$.

An ideal Bayesian observer selects the estimate that minimizes the expected loss, given the posterior and a loss function. We assume a least-squares loss function. Then, the optimal estimate $\hat{v}$ is the mean of the posterior in Equation (1). It is easy to see why this model of a Bayesian observer is consistent with the fact that perceived speed decreases with contrast. The width of the likelihood varies inversely with the accuracy of the measurements performed by the observer, which presumably decreases with decreasing contrast due to a decreasing signal-to-noise ratio. As illustrated in Figure 1, the shift in perceived speed towards slow velocities grows with the width of the likelihood, and thus a Bayesian model can qualitatively explain the psychophysical results [1].

## 1.2 Two Alternative Forced Choice Experiment

We would like to examine perceived speeds under a wide range of conditions in order to constrain a Bayesian model. Unfortunately, perceived speed is an internal variable, and it is not obvious how to design an experiment that would allow subjects to express it directly [1]. Perceived speed can only be accessed indirectly by asking the subject to *compare* the speed of two stimuli. For a given trial, an ideal Bayesian observer in such a two-alternative forced choice (2AFC) experimental paradigm simply decides on the basis of the two trial estimates $\hat{v}_1$ (stimulus1) and $\hat{v}_2$ (stimulus2) which stimulus moves faster. Each estimate $\hat{v}$ is based on a particular measurement $m$. For a given stimulus with speed $v$, an ideal Bayesian observer will produce a *distribution of estimates* $p(\hat{v}|v)$ because $m$ is noisy. Over trials, the observers behavior can be described by classical signal detection theory based on the distributions of the estimates, hence *e.g.* the probability of perceiving stimulus2 moving

faster than stimulus1 is given as the cumulative probability

$$P_{cum}(\hat{v}_2 > \hat{v}_1) = \int_0^\infty p(\hat{v}_2|v_2) \int_0^{\hat{v}_2} p(\hat{v}_1|v_1) \, d\hat{v}_1 \, d\hat{v}_2 \qquad (2)$$

$P_{cum}$ describes the full psychometric curve. Figure 2b illustrates the measured psychometric curve and its fit from such an experimental situation.

## 2   Experimental Methods

We measured matching speeds ($P_{cum} = 0.5$) and thresholds ($P_{cum} = 0.875$) in a 2AFC speed discrimination task. Subjects were presented simultaneously with two circular patches of horizontally drifting sine-wave gratings for the duration of one second (Figure 2a). Patches were 3deg in diameter, and were displayed at 6deg eccentricity to either side of a fixation cross. The stimuli had an identical spatial frequency of 1.5 cycle/deg. One stimulus was considered to be the reference stimulus having one of two different contrast values ($c_1$=[0.075 0.5]) and one of five different speed values ($u_1$=[1 2 4 8 12] deg/sec) while the second stimulus (test) had one of five different contrast values ($c_2$=[0.05 0.1 0.2 0.4 0.8]) and a varying speed that was determined by an interleaved staircase procedure. For each condition there were 96 trials. Conditions were randomly interleaved, including a random choice of stimulus identity (test vs. reference) and motion direction (right vs. left). Subjects were asked to fixate during stimulus presentation and select the faster moving stimulus. The threshold experiment differed only in that auditory feedback was given to indicate the correctness of their decision. This did not change the outcome of the experiment but increased significantly the quality of the data and thus reduced the number of trials needed.

## 3   Analysis

With the data from the speed discrimination experiments we could in principal apply a parametric fit using Equation (2) to derive the prior and the likelihood, but the optimization is difficult, and the fit might not be well constrained given the amount of data we have obtained. The problem becomes much more tractable given the following weak assumptions:

- We consider the prior to be relatively smooth.
- We assume that the measurement $m$ is corrupted by additive Gaussian noise with a variance whose dependence on stimulus speed and contrast is *separable*.
- We assume that there is a mapping function $f(v) : v \rightarrow v_m$ that maps $v$ into the space of $m$ (m-space). In that space, the likelihood is convolutional *i.e.* the noise in the measurement directly defines the width of the likelihood.

These assumptions allow us to relate the psychophysical data to our probabilistic model in a simple way. The following analysis is in the m-space. The point of subjective equality ($P_{cum} = 0.5$) is defined as where the expected values of the speed estimates are equal. We write

$$\begin{aligned} \mathcal{E}\langle \hat{v}_{m,1} \rangle &= \mathcal{E}\langle \hat{v}_{m,2} \rangle \\ v_{m,1} - \mathcal{E}\langle \mu_1 \rangle &= v_{m,2} - \mathcal{E}\langle \mu_2 \rangle \end{aligned} \qquad (3)$$

where $\mathcal{E}\langle \mu \rangle$ is the expected shift of the perceived speed compared to the veridical speed. For the discrimination threshold experiment, above assumptions imply that the variance $\mathrm{var}\langle \hat{v}_m \rangle$ of the speed estimates $\hat{v}_m$ is equal for both stimuli. Then, (2) predicts that the discrimination threshold is proportional to the standard deviation, thus

$$v_{m,2} - v_{m,1} = \gamma \sqrt{\mathrm{var}\langle \hat{v}_m \rangle} \qquad (4)$$

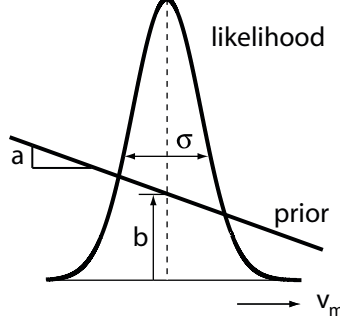

Figure 3: *Piece-wise approximation* We perform a parametric fit by assuming the prior to be piece-wise linear and the likelihood to be LogNormal (Gaussian in the m-space).

where $\gamma$ is a constant that depends on the threshold criterion $P_{cum}$ and the exact shape of $p(\hat{v}_m|v_m)$.

### 3.1  Estimating the prior and likelihood

In order to extract the prior and the likelihood of our model from the data, we have to find a generic local form of the prior and the likelihood and relate them to the mean and the variance of the speed estimates. As illustrated in Figure 3, we assume that the likelihood is Gaussian with a standard deviation $\sigma(c, v_m)$. Furthermore, the prior is assumed to be well-approximated by a first-order Taylor series expansion over the velocity ranges covered by the likelihood. We parameterize this linear expansion of the prior as $p(v_m) = av_m + b$.

We now can derive a posterior for this local approximation of likelihood and prior and then define the perceived speed shift $\mu(m)$. The posterior can be written as

$$p(v_m|m) = \frac{1}{\alpha}\, p(m|v_m)p(v_m) = \frac{1}{\alpha}\,[\exp(-\frac{v_m^2}{2\sigma(c, v_m)^2})(av_m + b)] \tag{5}$$

where $\alpha$ is the normalization constant

$$\alpha = \int_{-\infty}^{\infty} p(m|v_m)p(v_m)dv_m = \frac{b}{2}\sqrt{\pi 2\sigma(c, v_m)^2} \tag{6}$$

We can compute $\mu(m)$ as the first order moment of the posterior for a given $m$. Exploiting the symmetries around the origin, we find

$$\mu(m) = \int_{-\infty}^{\infty} vp(v_m|m)dv_m \equiv \frac{a(m)}{b(m)}\sigma(c, v_m)^2 \tag{7}$$

The expected value of $\mu(m)$ is equal to the value of $\mu$ at the expected value of the measurement $m$ (which is the stimulus velocity $v_m$), thus

$$\mathcal{E}\langle\mu\rangle = \mu(m)|_{m=v_m} = \frac{a(v_m)}{b(v_m)}\sigma(c, v_m)^2 \tag{8}$$

Similarly, we derive $\mathrm{var}\langle\hat{v}_m\rangle$. Because the estimator is deterministic, the variance of the estimate only depends on the variance of the measurement $m$. For a given stimulus, the variance of the estimate can be well approximated by

$$\begin{aligned}
\mathrm{var}\langle\hat{v}_m\rangle &= \mathrm{var}\langle m\rangle(\frac{\partial\hat{v}_m(m)}{\partial m}|_{m=v_m})^2 \tag{9}\\
&= \mathrm{var}\langle m\rangle(1 - \frac{\partial\mu(m)}{\partial m}|_{m=v_m})^2 \approx \mathrm{var}\langle m\rangle
\end{aligned}$$

Under the assumption of a locally smooth prior, the perceived velocity shift remains locally constant. The variance of the perceived speed $\hat{v}_m$ becomes equal to the variance of the measurement $m$, which is the variance of the likelihood (in the m-space), thus

$$\mathrm{var}\langle \hat{v}_m \rangle = \sigma(c, v_m)^2 \qquad (10)$$

With (3) and (4), above derivations provide a simple dependency of the psychophysical data to the local parameters of the likelihood and the prior.

### 3.2 Choosing a Logarithmic speed representation

We now want to choose the appropriate mapping function $f(v)$ that maps $v$ to the m-space. We define the m-space as the space in which the likelihood is Gaussian with a speed-independent width. We have shown that discrimination threshold is proportional to the width of the likelihood (4), (10). Also, we know from the psychophysics literature that visual speed discrimination approximately follows a Weber-Fechner law [11, 12], thus that the discrimination threshold increases roughly proportional with speed and so would the likelihood. A logarithmic speed representation would be compatible with the data and our choice of the likelihood. Hence, we transform the linear speed-domain $v$ into a normalized logarithmic domain according to

$$v_m = f(v) = \ln\left(\frac{v + v_0}{v_0}\right) \qquad (11)$$

where $v_0$ is a small normalization constant. The normalization is chosen to account for the expected deviation of equal variance behavior at the low end. Surprisingly, it has been found that neurons in the Medial Temporal area (Area MT) of macaque monkeys have speed-tuning curves that are very well approximated by Gaussians of constant width in above normalized logarithmic space [13]. These neurons are known to play a central role in the representation of motion. It seems natural to assume that they are strongly involved in tasks such as our performed psychophysical experiments.

## 4 Results

Figure 4 shows the contrast dependent shift of speed perception and the speed discrimination threshold data for two subjects. Data points connected with a dashed line represent the relative matching speed $(v_2/v_1)$ for a particular contrast value $c_2$ of the test stimulus as a function of the speed of the reference stimulus. Error bars are the empirical standard deviation of fits to bootstrapped samples of the data. Clearly, low contrast stimuli are perceived to move slower. The effect, however, varies across the tested speed range and tends to become smaller for higher speeds. The relative discrimination thresholds for two different contrasts as a function of speed show that the Weber-Fechner law holds only approximately. The data are in good agreement with other data from the psychophysics literature [1, 11, 8].

For each subject, data from both experiments were used to compute a parametric least-squares fit according to (3), (4), (7), and (10). In order to *test* the assumption of a LogNormal likelihood we allowed the standard deviation to be dependent on contrast and speed, thus $\sigma(c, v_m) = g(c)h(v_m)$. We split the speed range into six bins (subject2: five) and parameterized $h(v_m)$ and the ratio $a/b$ accordingly. Similarly, we parameterized $g(c)$ for the seven contrast values. The resulting fits are superimposed as bold lines in Figure 4.

Figure 5 shows the fitted parametric values for $g(c)$ and $h(v)$ (plotted in the linear domain), and the reconstructed prior distribution $p(v)$ transformed back to the linear domain. The approximately constant values for $h(v)$ provide evidence that a LogNormal distribution is an appropriate functional description of the likelihood. The resulting values for $g(c)$ suggest for the likelihood width a roughly exponential decaying dependency on contrast with strong saturation for higher contrasts.

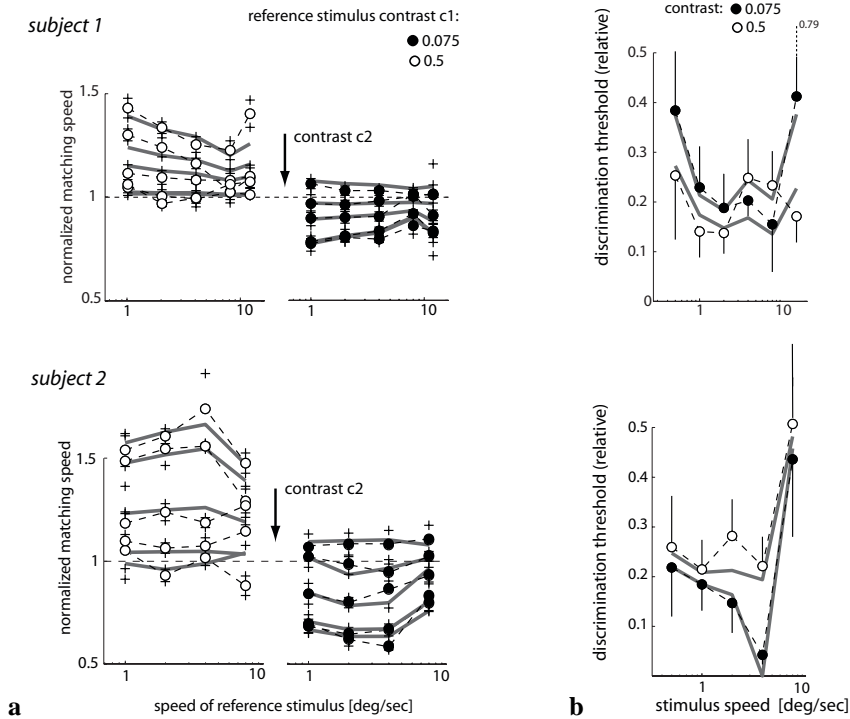

Figure 4: *Speed discrimination data for two subjects.* a) The relative matching speed of a test stimulus with different contrast levels ($c_2$=[0.05 0.1 0.2 0.4 0.8]) to achieve subjective equality with a reference stimulus (two different contrast values $c_1$). b) The relative discrimination threshold for two stimuli with equal contrast ($c_{1,2}$=[0.075 0.5]).

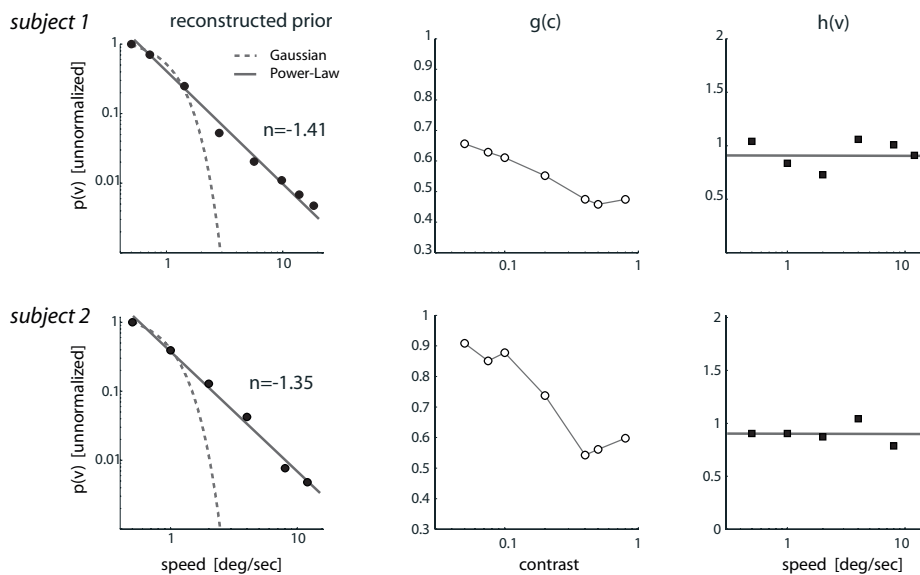

Figure 5: *Reconstructed prior distribution and parameters of the likelihood function.* The reconstructed prior for both subjects show much heavier tails than a Gaussian (dashed fit), approximately following a power-law function with exponent $n \approx -1.4$ (bold line).

# 5 Conclusions

We have proposed a probabilistic framework based on a Bayesian ideal observer and standard signal detection theory. We have derived a likelihood function and prior distribution for the estimator, with a fairly conservative set of assumptions, constrained by psychophysical measurements of speed discrimination and matching. The width of the resulting likelihood is nearly constant in the logarithmic speed domain, and decreases approximately exponentially with contrast. The prior expresses a preference for slower speeds, and approximately follows a power-law distribution, thus has much heavier tails than a Gaussian.

It would be interesting to compare the here derived prior distributions with measured true distributions of local image velocities that impinge on the retina. Although a number of authors have measured the spatio-temporal structure of natural images [14, e.g. ], it is clearly difficult to extract therefrom the true prior distribution because of the feedback loop formed through movements of the body, head and eyes.

### Acknowledgments

The authors thank all subjects for their participation in the psychophysical experiments.

## Footnotes

[1]Although see [10] for an example of determining and even changing the prior of a Bayesian model for a sensorimotor task, where the estimates are more directly accessible.

# References

[1] P. Thompson. Perceived rate of movement depends on contrast. *Vision Research*, 22:377–380, 1982.

[2] L.S. Stone and P. Thompson. Human speed perception is contrast dependent. *Vision Research*, 32(8):1535–1549, 1992.

[3] A. Yuille and N. Grzywacz. A computational theory for the perception of coherent visual motion. *Nature*, 333(5):71–74, May 1988.

[4] Alan Stocker. *Constraint Optimization Networks for Visual Motion Perception - Analysis and Synthesis*. PhD thesis, Dept. of Physics, Swiss Federal Institute of Technology, Zürich, Switzerland, March 2002.

[5] Eero Simoncelli. *Distributed analysis and representation of visual motion*. PhD thesis, MIT, Dept. of Electrical Engineering, Cambridge, MA, 1993.

[6] Y. Weiss, E. Simoncelli, and E. Adelson. Motion illusions as optimal percept. *Nature Neuroscience*, 5(6):598–604, June 2002.

[7] D.M. Green and J.A. Swets. *Signal Detection Theory and Psychophysics*. Wiley, New York, 1966.

[8] F. Hürlimann, D. Kiper, and M. Carandini. Testing the Bayesian model of perceived speed. *Vision Research*, 2002.

[9] Y. Weiss and D.J. Fleet. *Probabilistic Models of the Brain*, chapter Velocity Likelihoods in Biological and Machine Vision, pages 77–96. Bradford, 2002.

[10] K. Koerding and D. Wolpert. Bayesian integration in sensorimotor learning. *Nature*, 427(15):244–247, January 2004.

[11] Leslie Welch. The perception of moving plaids reveals two motion-processing stages. *Nature*, 337:734–736, 1989.

[12] S. McKee, G. Silvermann, and K. Nakayama. Precise velocity discrimination despite random variations in temporal frequency and contrast. *Vision Research*, 26(4):609–619, 1986.

[13] C.H. Anderson, H. Nover, and G.C. DeAngelis. Modeling the velocity tuning of macaque MT neurons. *Journal of Vision/VSS abstract*, 2003.

[14] D.W. Dong and J.J. Atick. Statistics of natural time-varying images. *Network: Computation in Neural Systems*, 6:345–358, 1995.
